# Tempering Backpropagation Networks: Not All Weights are Created Equal

**Nicol N. Schraudolph**
EVOTEC BioSystems GmbH
Grandweg 64
22529 Hamburg, Germany
*nici@evotec.de*

**Terrence J. Sejnowski**
Computational Neurobiology Lab
The Salk Institute for Biol. Studies
San Diego, CA 92186-5800, USA
*terry@salk.edu*

## Abstract

Backpropagation learning algorithms typically collapse the network's structure into a single vector of weight parameters to be optimized. We suggest that their performance may be improved by utilizing the structural information instead of discarding it, and introduce a framework for "tempering" each weight accordingly.

In the tempering model, activation and error signals are treated as approximately independent random variables. The characteristic scale of weight changes is then matched to that of the residuals, allowing structural properties such as a node's fan-in and fan-out to affect the local learning rate and backpropagated error. The model also permits calculation of an upper bound on the global learning rate for batch updates, which in turn leads to different update rules for bias *vs.* non-bias weights.

This approach yields hitherto unparalleled performance on the family relations benchmark, a deep multi-layer network: for both batch learning with momentum and the *delta-bar-delta* algorithm, convergence at the optimal learning rate is sped up by more than an order of magnitude.

## 1  Introduction

Although neural networks are structured graphs, learning algorithms typically view them as a single vector of parameters to be optimized. All information about a network's architecture is thus discarded in favor of the presumption of an *isotropic* weight space — the notion that *a priori* all weights in the network are created equal. This serves to decouple the learning process from network design and makes a large body of function optimization techniques directly applicable to backpropagation learning.

But what if the discarded structural information holds valuable clues for efficient weight optimization? Adaptive step size and second-order gradient techniques (Battiti, 1992) may

recover some of it, at considerable computational expense. *Ad hoc* attempts to incorporate structural information such as the fan-in (Plaut et al., 1986) into local learning rates have become a familiar part of backpropagation lore; here we derive a more comprehensive framework — which we call *tempering* — and demonstrate its effectiveness.

Tempering is based on modeling the activities and error signals in a backpropagation network as independent random variables. This allows us to calculate activity- and weight-invariant upper bounds on the effect of synchronous weight updates on a node's activity. We then derive appropriate local step size parameters by relating this maximal change in a node's activity to the characteristic scale of its residual through a global learning rate.

Our subsequent derivation of an upper bound on the global learning rate for batch learning suggests that the d.c. component of the error signal be given special treatment. Our experiments show that the resulting method of *error shunting* allows the global learning rate to approach its predicted maximum, for highly efficient learning performance.

## 2 Local Learning Rates

Consider a neural network with feedforward activation given by

$$x_j = f_j(y_j), \quad y_j = \sum_{i \in A_j} x_i \, w_{ij}, \tag{1}$$

where $A_j$ denotes the set of *anterior* nodes feeding directly into node $j$, and $f_j$ is a nonlinear (typically sigmoid) activation function. We imply that nodes are activated in the appropriate sequence, and that some have their values clamped so as to represent external inputs.

With a local learning rate of $\eta_j$ for node $j$, gradient descent in an objective function $E$ produces the weight update

$$\Delta w_{ij} = \eta_j \, \delta_j \, x_i, \quad \text{where} \quad \delta_j \equiv \frac{\partial E}{\partial y_j}. \tag{2}$$

Linearizing $f_j$ around $y_j$ approximates the resultant change in activation $x_j$ as

$$\Delta x_j \approx f_j'(y_j) \sum_{i \in A_j} x_i \, \Delta w_{ij} = \eta_j \, \delta_j \, f_j'(y_j) \sum_{i \in A_j} x_i^2. \tag{3}$$

Our goal is to put the scale of $\Delta x_j$ in relation to that of the error signal $\delta_j$. Specifically, when averaged over many training samples, we want the change in output activity of each node in response to each pattern limited to a certain proportion — given by the global learning rate $\eta$ — of its residual. We achieve this by relating the *variation* of $\Delta x_j$ over the training set to that of the error signal:

$$(\forall j) \quad \langle \Delta x_j^2 \rangle \leq \eta^2 \langle \delta_j^2 \rangle, \tag{4}$$

where $\langle \cdot \rangle$ denotes averaging over training samples. Formally, this approach may be interpreted as a diagonal approximation of the inverse Fischer information matrix (Amari, 1995). We implement (4) by deriving an upper bound for the left-hand side which is then equated with the right-hand side. Replacing the activity-dependent slope of $f_j$ by its maximum value

$$s(f_j) \equiv \max_u |f_j'(u)| \tag{5}$$

and assuming that there are no correlations[1] between inputs $x_i$ and error $\delta_j$, we obtain

$$\langle \Delta x_j^2 \rangle \leq \eta_j^2 \, s(f_j)^2 \langle \delta_j^2 \rangle \xi_j \tag{6}$$

from (3), provided that

$$\xi_j \geq \xi_j^* \equiv \left\langle \left[ \sum_{i \in A_j} x_i^2 \right]^2 \right\rangle . \tag{7}$$

We can now satisfy (4) by setting the local learning rate to

$$\eta_j \equiv \frac{\eta}{s(f_j)\sqrt{\xi_j}} . \tag{8}$$

There are several approaches to computing an upper bound $\xi_j$ on the total squared input power $\xi_j^*$. One option would be to calculate the latter empirically during training, though this raises sampling and stability issues. For external inputs we may precompute $\xi_j^*$ or derive an upper bound based on prior knowledge of the training data. For inputs from other nodes in the network we assume independence and derive $\xi_j$ from the range of their activation functions:

$$\xi_j = \sum_{i \in A_j} p(f_i)^2 , \quad \text{where} \quad p(f_i) \equiv \max_u f_i(u)^2 . \tag{9}$$

Note that when all nodes use the same activation function $f$, we obtain the well-known $\sqrt{fan\text{-}in}$ heuristic (Plaut et al., 1986) as a special case of (8).

## 3   Error Backpropagation

In deriving local learning rates above we have tacitly used the error signal as a stand-in for the residual proper, *i.e.* the distance to the target. For output nodes we can scale the error to never exceed the residual:

$$\delta_j = \frac{1}{\phi_j} \frac{\partial E}{\partial y_j} , \quad \text{where} \quad \phi_j \equiv \max_{y_j} \left| f_j'(y_j) \frac{\partial^2 E}{\partial f_j(y_j)^2} \right| . \tag{10}$$

Note that for the conventional quadratic error this simplifies to $\phi_j = s(f_j)$. What about the remainder of the network? Unlike (Krogh et al., 1990), we do not wish to prescribe definite targets (and hence residuals) for hidden nodes. Instead we shall use our bounds and independence arguments to scale backpropagated error signals to roughly appropriate magnitude. For this purpose we introduce an attenuation coefficient $a_i$ into the error back-propagation equation:

$$\delta_i = a_i f_i'(y_i) \sum_{j \in P_i} w_{ij} \delta_j , \tag{11}$$

where $P_i$ denotes the set of *posterior* nodes fed directly from node $i$. We posit that the appropriate variation for $\delta_i$ be no more than the weighted average of the variation of back-propagated errors:

$$\langle \delta_i^2 \rangle \leq \frac{1}{|P_i|} \sum_{j \in P_i} w_{ij}^2 \langle \delta_j^2 \rangle \tag{12}$$

whereas, assuming independence between the $\delta_j$ and replacing the slope of $f_i$ by its maximum value, (11) gives us

$$\langle \delta_i^2 \rangle \leq a_i^2 s(f_i)^2 \sum_{j \in P_i} w_{ij}^2 \langle \delta_j^2 \rangle . \tag{13}$$

Again we equate the right-hand sides of both inequalities to satisfy (12), yielding

$$a_i \equiv \frac{1}{s(f_i)\sqrt{|P_i|}} . \tag{14}$$

Note that the incorporation of the weights into (12) is *ad hoc*, as we have no *a priori* reason to scale a node's step size in proportion to the size of its vector of outgoing weights. We have chosen (12) simply because it produces a weight-invariant value for the attenuation coefficient. The scale of the backpropagated error could be controlled more rigorously, at the expense of having to recalculate $a_i$ after each weight update.

## 4 Global Learning Rate

We now derive the appropriate global learning rate for the batch weight update

$$\widehat{\Delta} w_{ij} \equiv \eta_j \sum_{t \in T} \delta_j(t)\, x_i(t) \tag{15}$$

over a non-redundant training sample $T$. Assuming independent and zero-mean residuals, we then have

$$\widehat{\Delta} x_j^2 = |T| \left\langle \Delta x_j^2 \right\rangle \leq |T| \eta^2 \left\langle \delta_j^2 \right\rangle \tag{16}$$

by virtue of (4). Under these conditions we can ensure

$$\widehat{\Delta} x_j^2 \leq \left\langle \delta_j^2 \right\rangle, \tag{17}$$

*i.e.* that the variation of the batch weight update does not exceed that of the residual, by using a global learning rate of

$$\eta \leq \eta^* \equiv 1/\sqrt{|T|}. \tag{18}$$

Even when redundancy in the training set forces us to use a lower rate, knowing the upper bound $\eta^*$ effectively allows an educated guess at $\eta$, saving considerable time in practice.

## 5 Error Shunting

It remains to deal with the assumption made above that the residuals be zero-mean, *i.e.* that $\langle \delta_j \rangle = 0$. Any d.c. component in the error requires a learning rate inversely proportional to the batch size — far below $\eta^*$, the rate permissible for zero-mean residuals. This suggests handling the d.c. component of error signals separately. This is the proper job of the bias weight, so we update it accordingly:

$$\widehat{\Delta} w_{oj} = \langle \delta_j \rangle / s(f_j). \tag{19}$$

In order to allow learning at rates close to $\eta^*$ for all other weights, their error signals are then centered by subtracting the mean:

$$(\forall i \neq 0) \quad \widehat{\Delta} w_{ij} = \eta_j \sum_{t \in T} \left( \delta_j(t) - \langle \delta_j \rangle \right) x_i(t) \tag{20}$$

$$= \eta_j \left( \sum_{t \in T} \delta_j(t)\, x_i(t) - \langle x_i \rangle \sum_{t \in T} \delta_j(t) \right). \tag{21}$$

Note that both sums in (21) must be collected in batch implementations of backpropagation anyway — the only additional statistic required is the average input activity $\langle x_i \rangle$. Indeed for batch update centering errors is equivalent to centering inputs, which is known to assist learning by removing a large eigenvalue of the Hessian (LeCun et al., 1991). We expect online implementations to perform best when *both* input and error signals are centered so as to improve the stochastic approximation.

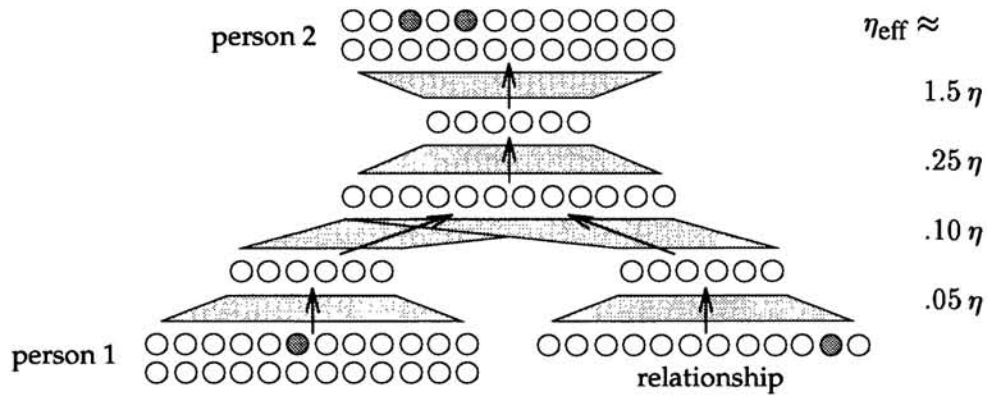

person 2

$\eta_{\text{eff}} \approx$

$1.5\,\eta$

$.25\,\eta$

$.10\,\eta$

$.05\,\eta$

person 1                                                relationship

Figure 1: Backpropagation network for learning family relations (Hinton, 1986).

## 6   Experimental Setup

We tested these ideas on the family relations task (Hinton, 1986): a backpropagation network is given examples of a family member and relationship as input, and must indicate on its output which family members fit the relational description according to an underlying family tree. Its architecture (Figure 1) consists of a central *association* layer of hidden units surrounded by three *encoding* layers that act as informational bottlenecks, forcing the network to make the deep structure of the data explicit.

The input is presented to the network in a canonical local encoding: for any given training example, exactly one input in each of the two input layers is active. On account of the always active bias input, the squared input power for tempering at these layers is thus $\xi^* = 4$. Since the output uses the same local code, only one or two targets at a time will be active; we therefore do not attenuate error signals in the immediately preceding layer. We use cross-entropy error and the logistic squashing function $(1 + e^{-y})^{-1}$ at the output (giving $\phi = 1$) but prefer the hyperbolic tangent for hidden units, with $p(\tanh) = s(\tanh) = 1$.

To illustrate the impact of tempering on this architecture we translate the combined effect of local learning rate and error attenuation into an *effective* learning rate[2] for each layer, shown on the right in Figure 1. We observe that effective learning rates are largest near the output and decrease towards the input due to error attenuation. Contrary to textbook opinion (LeCun, 1993; Haykin, 1994, page 162) we find that such unequal step sizes are in fact the key to efficient learning here. We suspect that the logistic squashing function may owe its popularity largely to the error attenuation side-effect inherent in its maximum slope of $^{1}/_{4}$.

We expect tempering to be applicable to a variety of backpropagation learning algorithms; here we present first results for batch learning with momentum and the *delta-bar-delta* rule (Jacobs, 1988). Both algorithms were tested under three conditions: conventional, tempered (as described in Sections 2 and 3), and tempered with error shunting. All experiments were performed with a customized simulator based on Xerion 3.1.[3]

For each condition the global learning rate $\eta$ was empirically optimized (to single-digit precision) for fastest reliable learning performance, as measured by the sum of empirical mean and standard deviation of epochs required to reach a given low value of the cost function. All other parameters were held invariant across experiments; their values (shown in Table 1) were chosen in advance so as not to bias the results.

| Parameter | Value | Parameter | Value |
|---|---|---|---|
| training set size (= epoch) | 100 | zero-error radius around target | 0.2 |
| momentum parameter | 0.9 | acceptable error & weight cost | 1.0 |
| uniform initial weight range | $\pm 0.3$ | delta-bar-delta gain increment | 0.1 |
| weight decay rate per epoch | $10^{-4}$ | delta-bar-delta gain decrement | 0.9 |

Table 1: Invariant parameter settings for our experiments.

## 7   Experimental Results

Table 2 lists the empirical mean and standard deviation (over ten restarts) of the number of epochs required to learn the family relations task under each condition, and the optimal learning rate that produced this performance. Training times for conventional backpropagation are quite long; this is typical for deep multi-layer networks. For comparison, Hinton reports around 1,500 epochs on this problem when both learning rate and momentum have been optimized (personal communication). Much faster convergence — though to a far looser criterion — has recently been observed for online algorithms (O'Reilly, 1996).

Tempering, on the other hand, is seen here to speed up two batch learning methods by almost an order of magnitude. It reduces not only the average training time but also its coefficient of variation, indicating a more reliable optimization process. Note that tempering makes simple batch learning with momentum run about twice as fast as the delta-bar-delta algorithm. This is remarkable since delta-bar-delta uses online measurements to continually adapt the learning rate for each individual weight, whereas tempering merely prescales it based on the network's architecture. We take this as evidence that tempering establishes appropriate local step sizes upfront that delta-bar-delta must discover empirically.

This suggests that by using tempering to set the initial (equilibrium) learning rates for delta-bar-delta, it may be possible to reap the benefits of both prescaling and adaptive step size control. Indeed Table 2 confirms that the respective speedups due to tempering and delta-bar-delta multiply when the two approaches are combined in this fashion. Finally, the addition of error shunting increases learning speed yet further by allowing the global learning rate to be brought close to the maximum of $\eta^* = 0.1$ that we would predict from (18).

## 8   Discussion

In our experiments we have found tempering to dramatically improve speed and reliability of learning. More network architectures, data sets and learning algorithms will have to be "tempered" to explore the general applicability and limitations of this approach; we also hope to extend it to recurrent networks and online learning. Error shunting has proven useful in facilitating of near-maximal global learning rates for rapid optimization.

| Algorithm<br>Condition | batch & momentum | | | delta-bar-delta | | |
|---|---|---|---|---|---|---|
| | $\eta =$ | mean | st.d. | $\eta =$ | mean | st.d. |
| conventional | $3 \cdot 10^{-3}$ | $2438 \pm 1153$ | | $3 \cdot 10^{-4}$ | $696 \pm 218$ | |
| with tempering | $1 \cdot 10^{-2}$ | $339 \pm 95.0$ | | $3 \cdot 10^{-2}$ | $89.6 \pm 11.8$ | |
| tempering & shunting | $4 \cdot 10^{-2}$ | $142 \pm 27.1$ | | $9 \cdot 10^{-2}$ | $61.7 \pm 8.1$ | |

Table 2: Epochs required to learn the family relations task.

Although other schemes may speed up backpropagation by comparable amounts, our approach has some unique advantages. It is computationally cheap to implement: local learning and error attenuation rates are invariant with respect to network weights and activities and thus need to be recalculated only when the network architecture is changed.

More importantly, even advanced gradient descent methods typically retain the isotropic weight space assumption that we improve upon; one would therefore expect them to benefit from tempering as much as delta-bar-delta did in the experiments reported here. For instance, tempering could be used to set non-isotropic model-trust regions for conjugate and second-order gradient descent algorithms.

Finally, by restricting ourselves to fixed learning rates and attenuation factors for now we have arrived at a simplified method that is likely to leave room for further improvement. Possible refinements include taking weight vector size into account when attenuating error signals, or measuring quantities such as $\langle \delta^2 \rangle$ online instead of relying on invariant upper bounds. How such adaptive tempering schemes will compare to and interact with existing techniques for efficient backpropagation learning remains to be explored.

## Acknowledgements

We would like to thank Peter Dayan, Rich Zemel and Jenny Orr for being instrumental in discussions that helped shape this work. Geoff Hinton not only offered invaluable comments, but is the source of both our simulator and benchmark problem. N. Schraudolph received financial support from the McDonnell-Pew Center for Cognitive Neuroscience in San Diego, and the Robert Bosch Stiftung GmbH.

## Footnotes

[1] Note that such correlations are minimized by the local weight update.

[2]This is possible only for strictly layered networks, *i.e.* those with no shortcut (or "skip-through") connections between topologically non-adjacent layers.

[3]At the time of writing, the Xerion neural network simulator and its successor UTS are available by anonymous file transfer from ai.toronto.edu, directory pub/xerion.

## References

Amari, S.-I. (1995). Learning and statistical inference. In Arbib, M. A., editor, *The Handbook of Brain Theory and Neural Networks*, pages 522–526. MIT Press, Cambridge.

Battiti, T. (1992). First- and second-order methods for learning: Between steepest descent and Newton's method. *Neural Computation*, 4(2):141–166.

Haykin, S. (1994). *Neural Networks: A Comprehensive Foundation*. Macmillan, New York.

Hinton, G. (1986). Learning distributed representations of concepts. In *Proceedings of the Eighth Annual Conference of the Cognitive Science Society*, pages 1–12, Amherst 1986. Lawrence Erlbaum, Hillsdale.

Jacobs, R. (1988). Increased rates of convergence through learning rate adaptation. *Neural Networks*, 1:295–307.

Krogh, A., Thorbergsson, G., and Hertz, J. A. (1990). A cost function for internal representations. In Touretzky, D. S., editor, *Advances in Neural Information Processing Systems*, volume 2, pages 733–740, Denver, CO, 1989. Morgan Kaufmann, San Mateo.

LeCun, Y. (1993). Efficient learning & second-order methods. Tutorial given at the NIPS Conference, Denver, CO.

LeCun, Y., Kanter, I., and Solla, S. A. (1991). Second order properties of error surfaces: Learning time and generalization. In Lippmann, R. P., Moody, J. E., and Touretzky, D. S., editors, *Advances in Neural Information Processing Systems*, volume 3, pages 918–924, Denver, CO, 1990. Morgan Kaufmann, San Mateo.

O'Reilly, R. C. (1996). Biologically plausible error-driven learning using local activation differences: The generalized recirculation algorithm. *Neural Computation*, 8.

Plaut, D., Nowlan, S., and Hinton, G. (1986). Experiments on learning by back propagation. Technical Report CMU–CS–86–126, Department of Computer Science, Carnegie Mellon University, Pittsburgh, PA.